# Probabilistic Relational PCA

**Wu-Jun Li**       **Dit-Yan Yeung**
Dept. of Comp. Sci. and Eng.
Hong Kong University of Science and Technology
Hong Kong, China
{liwujun,dyyeung}@cse.ust.hk

**Zhihua Zhang**
School of Comp. Sci. and Tech.
Zhejiang University
Zhejiang 310027, China
zhzhang@cs.zju.edu.cn

## Abstract

One crucial assumption made by both principal component analysis (PCA) and probabilistic PCA (PPCA) is that the instances are independent and identically distributed (i.i.d.). However, this common i.i.d. assumption is unreasonable for relational data. In this paper, by explicitly modeling covariance between instances as derived from the relational information, we propose a novel probabilistic dimensionality reduction method, called *probabilistic relational PCA* (PRPCA), for relational data analysis. Although the i.i.d. assumption is no longer adopted in PRPCA, the learning algorithms for PRPCA can still be devised easily like those for PPCA which makes explicit use of the i.i.d. assumption. Experiments on real-world data sets show that PRPCA can effectively utilize the relational information to dramatically outperform PCA and achieve state-of-the-art performance.

## 1 Introduction

Using a low-dimensional embedding to summarize a high-dimensional data set has been widely used for exploring the structure in the data. The methods for discovering such low-dimensional embedding are often referred to as dimensionality reduction (DR) methods. Principal component analysis (PCA) [13] is one of the most popular DR methods with great success in many applications. As a more recent development, probabilistic PCA (PPCA) [21] provides a probabilistic formulation of PCA [13] based on a Gaussian latent variable model [1]. Compared with the original non-probabilistic derivation of PCA in [12], PPCA possesses a number of practical advantages. For example, PPCA can naturally deal with missing values in the data; the expectation-maximization (EM) algorithm [9] used to learn the parameters in PPCA may be more efficient for high-dimensional data; it is easy to generalize the single model in PPCA to the mixture model case; furthermore, PPCA as a probabilistic model can naturally exploit Bayesian methods [2].

Like many existing DR methods, both PCA and PPCA are based on some assumptions about the data. One assumption is that the data should be represented as feature vectors all of the same dimensionality. Data represented in this form are sometimes referred to as *flat data* [10]. Another one is the so-called i.i.d. assumption, which means that the instances are assumed to be independent and identically distributed (i.i.d.).

However, the data in many real-world applications, such as web pages and research papers, contain relations or links between (some) instances in the data in addition to the textual content information which is represented in the form of feature vectors. Data of this sort, referred to as *relational data*[1] [10, 20], can be found in such diverse application areas as web mining [3, 17, 23, 24], bioinformatics [22], social network analysis [4], and so on. *On one hand*, the link structure among instances

cannot be exploited easily when traditional DR methods such as PCA are applied to relational data. Very often, the useful relational information is simply discarded. For example, a citation/reference relation between two papers provides very strong evidence for them to belong to the same topic even though they may bear low similarity in their content due to the sparse nature of the bag-of-words representation, but the relational information is not exploited at all when applying PCA or PPCA. One possible use of the relational information in PCA or PPCA is to first convert the link structure into the format of flat data by extracting some additional features from the links. However, as argued in [10], this approach fails to capture some important structural information in the data. *On the other hand*, the i.i.d. assumption underlying PCA and PPCA is unreasonable for relational data. In relational data, the attributes of the connected (linked) instances are often *correlated* and the class label of one instance may have an influence on that of a linked instance. For example, in biology, interacting proteins are more likely to have the same biological functions than those without interaction. Therefore, PCA and PPCA, or more generally most existing DR methods based on the i.i.d. assumption, are not suitable for relational data analysis.

In this paper, a novel probabilistic DR method called *probabilistic relational PCA* (PRPCA) is proposed for relational data analysis. By explicitly modeling the covariance between instances as derived from the relational information, PRPCA seamlessly integrates relational information and textual content information into a unified probabilistic framework. Two learning algorithms, one based on a closed-form solution and the other based on an EM algorithm [9], are proposed to learn the parameters of PRPCA. Although the i.i.d. assumption is no longer adopted in PRPCA, the learning algorithms for PRPCA can still be devised easily like those for PPCA which makes explicit use of the i.i.d. assumption. Extensive experiments on real-world data sets show that PRPCA can effectively utilize the relational information to dramatically outperform PCA and achieve state-of-the-art performance.

## 2 Notation

We use boldface uppercase letters, such as $\mathbf{K}$, to denote matrices, and boldface lowercase letters, such as $\mathbf{z}$, to denote vectors. The $i$th row and the $j$th column of a matrix $\mathbf{K}$ are denoted by $\mathbf{K}_{i*}$ and $\mathbf{K}_{*j}$, respectively. $K_{ij}$ denotes the element at the $i$th row and $j$th column of $\mathbf{K}$. $z_i$ denotes the $i$th element of $\mathbf{z}$. $\mathbf{K}^T$ is the transpose of $\mathbf{K}$, and $\mathbf{K}^{-1}$ is the inverse of $\mathbf{K}$. $\mathbf{K} \succeq 0$ means that $\mathbf{K}$ is positive semi-definite (psd) and $\mathbf{K} \succ 0$ means that $\mathbf{K}$ is positive definite (pd). $\mathrm{tr}(\cdot)$ denotes the trace of a matrix and $\mathrm{etr}(\cdot) \triangleq \exp(\mathrm{tr}(\cdot))$. $\mathbf{P} \otimes \mathbf{Q}$ denotes the Kronecker product [11] of $\mathbf{P}$ and $\mathbf{Q}$. $|\cdot|$ denotes the determinant of a matrix. $\mathbf{I}_n$ is the identity matrix of size $n \times n$. $\mathbf{e}$ is a vector of 1s, the dimensionality of which depends on the context. We overload $\mathcal{N}(\cdot)$ for both multivariate normal distributions and matrix variate normal distributions [11]. $\langle \cdot \rangle$ denotes the expectation operation and $\mathrm{cov}(\cdot)$ denotes the covariance operation.

Note that in relational data, there exist both *content* and *link* observations. As in [21], $\{\mathbf{t}_n\}_{n=1}^N$ denotes a set of observed $d$-dimensional data (content) vectors, the $d \times q$ matrix $\mathbf{W}$ denotes the $q$ principal axes (or called factor loadings), $\boldsymbol{\mu}$ denotes the data sample mean, and $\mathbf{x}_n = \mathbf{W}^T(\mathbf{t}_n - \boldsymbol{\mu})$ denotes the corresponding $q$ principal components (or called latent variables) of $\mathbf{t}_n$. We further use the $d \times N$ matrix $\mathbf{T}$ to denote the content matrix with $\mathbf{T}_{*n} = \mathbf{t}_n$, and the $q \times N$ matrix $\mathbf{X}$ to denote the latent variables of $\mathbf{T}$ with $\mathbf{X}_{*n} = \mathbf{W}^T(\mathbf{t}_n - \boldsymbol{\mu})$. For relational data, the $N \times N$ matrix $\mathbf{A}$ denotes the adjacency (link) matrix of the $N$ instances. In this paper, we assume that the links are undirected. For those data with directed links, we will convert the directed links into undirected links which can keep the original physical meaning of the links. This will be described in detail in Section 4.1.1, and an example will be given in Section 5. Hence, $A_{ij} = 1$ if there exists a relation between instances $i$ and $j$, and otherwise $A_{ij} = 0$. Moreover, we always assume that there exist no self-links, i.e., $A_{ii} = 0$.

## 3 Probabilistic PCA

To set the stage for the next section which introduces our PRPCA model, we first briefly present the derivation for PPCA [21], which was originally based on (vector-based) multivariate normal distributions, from the perspective of matrix variate normal distributions [11].

If we use $\boldsymbol{\Upsilon}$ to denote the Gaussian noise process and assume that $\boldsymbol{\Upsilon}$ and the latent variable matrix $\mathbf{X}$ follow these distributions:
$$\boldsymbol{\Upsilon} \sim \mathcal{N}_{d,N}(\mathbf{0}, \sigma^2 \mathbf{I}_d \otimes \mathbf{I}_N), \qquad \mathbf{X} \sim \mathcal{N}_{q,N}(\mathbf{0}, \mathbf{I}_q \otimes \mathbf{I}_N), \tag{1}$$
we can express a generative model as follows: $\mathbf{T} = \mathbf{W}\mathbf{X} + \boldsymbol{\mu}\mathbf{e}^T + \boldsymbol{\Upsilon}$.

Based on some properties of matrix variate normal distributions in [11], we get the following results:
$$\mathbf{T} \mid \mathbf{X} \sim \mathcal{N}_{d,N}(\mathbf{W}\mathbf{X} + \boldsymbol{\mu}\mathbf{e}^T, \sigma^2 \mathbf{I}_d \otimes \mathbf{I}_N), \qquad \mathbf{T} \sim \mathcal{N}_{d,N}\left(\boldsymbol{\mu}\mathbf{e}^T, (\mathbf{W}\mathbf{W}^T + \sigma^2 \mathbf{I}_d) \otimes \mathbf{I}_N\right). \tag{2}$$
Let $\mathbf{C} = \mathbf{W}\mathbf{W}^T + \sigma^2 \mathbf{I}_d$. The corresponding log-likelihood of the observation matrix $\mathbf{T}$ is then
$$\mathcal{L} = \ln p(\mathbf{T}) = -\frac{N}{2}\Big[d\ln(2\pi) + \ln|\mathbf{C}| + \mathrm{tr}(\mathbf{C}^{-1}\mathbf{S})\Big], \tag{3}$$
where $\mathbf{S} = \frac{(\mathbf{T}-\boldsymbol{\mu}\mathbf{e}^T)(\mathbf{T}-\boldsymbol{\mu}\mathbf{e}^T)^T}{N} = \frac{\sum_{n=1}^{N}(\mathbf{T}_{*n}-\boldsymbol{\mu})(\mathbf{T}_{*n}-\boldsymbol{\mu})^T}{N}$. We can see that $\mathbf{S}$ is just the sample covariance matrix of the content observations. It is easy to see that this log-likelihood form is the same as that in [21]. Using matrix notations, the graphical model of PPCA based on matrix variate normal distributions is shown in Figure 1(a).

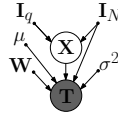 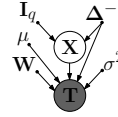

(a) Model of PPCA      (b) Model of PRPCA

Figure 1: Graphical models of PPCA and PRPCA, in which $\mathbf{T}$ is the observation matrix, $\mathbf{X}$ is the latent variable matrix, $\boldsymbol{\mu}$, $\mathbf{W}$ and $\sigma^2$ are the parameters to learn, and the other quantities are kept constant.

## 4 Probabilistic Relational PCA

PPCA assumes that all the observations are independent and identically distributed. Although this i.i.d. assumption can make the modeling process much simpler and has achieved great success in many traditional applications, this assumption is however very unreasonable for relational data [10]. In relational data, the attributes of connected (linked) instances are often *correlated*.

In this section, a probabilistic relational PCA model, called PRPCA, is proposed to integrate both the relational information and the content information seamlessly into a unified framework by eliminating the i.i.d. assumption. Based on our reformulation of PPCA using matrix variate notations as presented in the previous section, we can obtain PRPCA just by introducing some relatively simple (but very effective) modifications. A promising property is that the computation needed for PRPCA is as simple as that for PPCA even though we have eliminated the restrictive i.i.d. assumption.

### 4.1 Model Formulation

Assume that the latent variable matrix $\mathbf{X}$ has the following distribution:
$$\mathbf{X} \sim \mathcal{N}_{q,N}(\mathbf{0}, \mathbf{I}_q \otimes \boldsymbol{\Phi}). \tag{4}$$
According to Corollary 2.3.3.1 in [11], we can get $\mathrm{cov}(\mathbf{X}_{i*}) = \boldsymbol{\Phi}$ ($i \in \{1, \ldots, q\}$), which means that $\boldsymbol{\Phi}$ actually reflects the covariance between the instances. From (1), we can see that $\mathrm{cov}(\mathbf{X}_{i*}) = \mathbf{I}_N$ for PPCA, which also coincides with the i.i.d. assumption of PPCA.

Hence, to eliminate the i.i.d. assumption for relational data, one direct way is to use a non-identity covariance matrix $\boldsymbol{\Phi}$ for the distribution of $\mathbf{X}$ in (4). This $\boldsymbol{\Phi}$ should reflect the physical meaning (semantics) of the relations between instances, which will be discussed in detail later. Similarly, we can also change the $\mathbf{I}_N$ in (1) to $\boldsymbol{\Phi}$ for $\boldsymbol{\Upsilon}$ to eliminate the i.i.d. assumption for the noise process.

#### 4.1.1 Relational Covariance Construction

Because the covariance matrix $\boldsymbol{\Phi}$ in PRPCA is constructed from the relational information in the data, we refer to it as *relational covariance* here.

The goal of PCA and PPCA is to find those principal axes onto which the retained variance under projection is maximal [13, 21]. For one specific $\mathbf{X}$, the retained variance is $\mathrm{tr}[\mathbf{X}\mathbf{X}^T]$. If we rewrite $p(\mathbf{X})$ in (1) as $p(\mathbf{X}) = \frac{\exp\{\mathrm{tr}[-\frac{1}{2}\mathbf{X}\mathbf{X}^T]\}}{(2\pi)^{qN/2}} = \frac{\exp\{-\frac{1}{2}\mathrm{tr}[\mathbf{X}\mathbf{X}^T]\}}{(2\pi)^{qN/2}}$, we have the following observation:

**Observation 1** *For PPCA, the larger the retained variance of* $\mathbf{X}$, *i.e., the more* $\mathbf{X}$ *approaches the destination point, the lower is the probability density at* $\mathbf{X}$ *given by the prior.*

Here, the *destination point* refers to the point where the goal of PPCA is achieved, i.e., the retained variance is maximal. Moreover, we use the retained variance as a *measure* to define the gap between two different points. The smaller is the gap between the retained variance of two points, the more they approach each other.

Because the design principle of PRPCA is similar to that of PPCA, our working hypothesis here is that Observation 1 can also guide us to design the relational covariance of PRPCA. Its effectiveness will be empirically verified in Section 5.

In PRPCA, we assume that the attributes of two linked instances are positively correlated.[2] Under this assumption, the *ideal goal* of PRPCA should be to make the latent representations of two instances as close as possible if there exists a relation (link) between them. Hence, the *measure* to define the gap between two points refers to the closeness of the linked instances, i.e., the summation of the Euclidean distances between the linked instances. Based on Observation 1, the more $\mathbf{X}$ approaches the destination point, the lower should be the probability density at $\mathbf{X}$ given by the prior. Hence, under the latent space representation $\mathbf{X}$, the closer the linked instances are, the lower should be the probability density at $\mathbf{X}$ given by the prior. We will prove that if we set $\mathbf{\Phi} = \mathbf{\Delta}^{-1}$ where $\mathbf{\Delta} \triangleq \gamma \mathbf{I}_N + (\mathbf{I}_N + \mathbf{A})^T (\mathbf{I}_N + \mathbf{A})$ with $\gamma$ being typically a very small positive number to make $\mathbf{\Delta} \succ 0$, we can get an appropriate prior for PRPCA. Note that $A_{ij} = 1$ if there exists a relation between instances $i$ and $j$, and otherwise $A_{ij} = 0$. Because $\mathbf{A}^T = \mathbf{A}$, we can also express $\mathbf{\Delta}$ as $\mathbf{\Delta} = \gamma \mathbf{I}_N + (\mathbf{I}_N + \mathbf{A})(\mathbf{I}_N + \mathbf{A})$.

Let $\tilde{\mathbf{D}}$ denote a diagonal matrix whose diagonal elements $\tilde{D}_{ii} = \sum_j A_{ij}$. It is easy to prove that $(\mathbf{A}\mathbf{A})_{ii} = \tilde{D}_{ii}$. Let $\mathbf{B} = \mathbf{A}\mathbf{A} - \tilde{\mathbf{D}}$, which means that $B_{ij} = (\mathbf{A}\mathbf{A})_{ij}$ if $i \neq j$ and $B_{ii} = 0$. We can get $\mathbf{\Delta} = (1+\gamma)\mathbf{I}_N + 2\mathbf{A} + \mathbf{A}\mathbf{A} = (1+\gamma)\mathbf{I}_N + \tilde{\mathbf{D}} + (2\mathbf{A} + \mathbf{B})$. Because $B_{ij} = \sum_{k=1}^{N} A_{ik} A_{kj}$ for $i \neq j$, we can see that $B_{ij}$ is the number of paths, each with path length 2, from instance $i$ to instance $j$ in the original adjacency graph $\mathbf{A}$. Because the attributes of two linked instances are positively correlated, $B_{ij}$ actually reflects the degree of correlation between instance $i$ and instance $j$. Let us take the paper citation graph as an example to illustrate this. The existence of a citation relation between two papers often implies that they are about the same topic. If paper $i$ cites paper $k$ and paper $k$ cites paper $j$, it is highly likely that paper $i$ and paper $j$ are about the same topic. If there exists another paper $a \neq k$ linking both paper $i$ and paper $j$ as well, the confidence that paper $i$ and paper $j$ are about the same topic will increase. Hence, the larger $B_{ij}$ is, the stronger is the correlation between instance $i$ and instance $j$. Because $B_{ij} = \sum_{k=1}^{N} A_{ik} A_{kj} = \mathbf{A}_{*i}^T \mathbf{A}_{*j}$, $B_{ij}$ can also be seen as the similarity between the link vectors of instance $i$ and instance $j$. Therefore, $\mathbf{B}$ can be seen as a weight matrix (corresponding to a weight graph) derived from the original adjacency matrix $\mathbf{A}$, and $\mathbf{B}$ is also consistent with the physical meaning underlying $\mathbf{A}$.

Letting $\mathbf{G} = 2\mathbf{A} + \mathbf{B}$,[3] we can find that $\mathbf{G}$ actually combines the original graph reflected by $\mathbf{A}$ and the derived graph reflected by $\mathbf{B}$ to get a new graph, and puts a weight $2A_{ij} + B_{ij}$ on the edge between instance $i$ and instance $j$ in the new graph. The new weight graph reflected by $\mathbf{G}$ is also consistent with the physical meaning underlying $\mathbf{A}$. Letting $\mathbf{L} \triangleq \mathbf{D} - \mathbf{G}$, where $\mathbf{D}$ is a diagonal matrix whose diagonal elements $D_{ii} = \sum_j G_{ij}$ and $\mathbf{L}$ is called the Laplacian matrix [6] of $\mathbf{G}$, we can get $\mathbf{\Delta} = (1+\gamma)\mathbf{I}_N + \tilde{\mathbf{D}} + \mathbf{D} - \mathbf{L}$. If we define another diagonal matrix $\hat{\mathbf{D}} \triangleq (1+\gamma)\mathbf{I}_N + \tilde{\mathbf{D}} + \mathbf{D}$, we can get $\mathbf{\Delta} = \hat{\mathbf{D}} - \mathbf{L}$. Then we have

$$\mathrm{tr}[\mathbf{X}\mathbf{\Delta}\mathbf{X}^T] = \sum_{i=1}^{N} \hat{D}_{ii} \|\mathbf{X}_{*i}\|^2 - \frac{1}{2} \sum_{i=1}^{N} \sum_{j=1}^{N} G_{ij} \|\mathbf{X}_{*i} - \mathbf{X}_{*j}\|^2. \tag{5}$$

Letting $\mathbf{\Phi} = \mathbf{\Delta}^{-1}$, we can get $p(\mathbf{X}) = \frac{\exp\{\mathrm{tr}[-\frac{1}{2}\mathbf{X}\mathbf{\Delta}\mathbf{X}^T]\}}{(2\pi)^{qN/2}|\mathbf{\Delta}|^{-q/2}} = \frac{\exp\{-\frac{1}{2}\mathrm{tr}[\mathbf{X}\mathbf{\Delta}\mathbf{X}^T]\}}{(2\pi)^{qN/2}|\mathbf{\Delta}|^{-q/2}}$.

The first term $\sum_{i=1}^{N} \hat{D}_{ii}\|\mathbf{X}_{*i}\|^2$ in (5) can be treated as a measure of weighted variance of all the instances in the latent space. We can see that the larger $\hat{D}_{ii}$ is, the more weight will be put on instance $i$, which is reasonable because $\hat{D}_{ii}$ mainly reflects the degree of instance $i$ in the graph. It is easy to see that, for those latent representations having a fixed value of weighted variance $\sum_{i=1}^{N} \hat{D}_{ii}\|\mathbf{X}_{*i}\|^2$, the closer the latent representations of two linked entities are, the larger is their contribution to $\mathrm{tr}[\mathbf{X}\mathbf{\Delta}\mathbf{X}^T]$, and subsequently the less is their contribution to $p(\mathbf{X})$. This means that under the latent space representation $\mathbf{X}$, the closer the linked instances are, the lower is the probability density at $\mathbf{X}$ given by the prior. Hence, we can get an appropriate prior for $\mathbf{X}$ by setting $\mathbf{\Phi} = \mathbf{\Delta}^{-1}$ in (4).

### 4.1.2 Model

With the constructed relational covariance $\mathbf{\Phi}$, the generative model of PRPCA is defined as follows:

$$\mathbf{\Upsilon} \sim \mathcal{N}_{d,N}(\mathbf{0}, \sigma^2\mathbf{I}_d \otimes \mathbf{\Phi}), \quad \mathbf{X} \sim \mathcal{N}_{q,N}(\mathbf{0}, \mathbf{I}_q \otimes \mathbf{\Phi}), \quad \mathbf{T} = \mathbf{W}\mathbf{X} + \boldsymbol{\mu}\mathbf{e}^T + \mathbf{\Upsilon},$$

where $\mathbf{\Phi} = \mathbf{\Delta}^{-1}$.

We can further obtain the following results:

$$\mathbf{T} \mid \mathbf{X} \sim \mathcal{N}_{d,N}(\mathbf{W}\mathbf{X} + \boldsymbol{\mu}\mathbf{e}^T, \sigma^2\mathbf{I}_d \otimes \mathbf{\Phi}), \quad \mathbf{T} \sim \mathcal{N}_{d,N}\left(\boldsymbol{\mu}\mathbf{e}^T, (\mathbf{W}\mathbf{W}^T + \sigma^2\mathbf{I}_d) \otimes \mathbf{\Phi}\right). \quad (6)$$

The graphical model of PRPCA is illustrated in Figure 1(b), from which we can see that the difference between PRPCA and PPCA lies solely in the difference between $\mathbf{\Phi}$ and $\mathbf{I}_N$. Comparing (6) to (2), we can find that the observations of PPCA are sampled independently while those of PRPCA are sampled with correlation. In fact, PPCA may be seen as a degenerate case of PRPCA as detailed below in Remark 1:

**Remark 1** *When the i.i.d. assumption holds, i.e., all $A_{ij} = 0$, PRPCA degenerates to PPCA by setting $\gamma = 0$. Note that the only role that $\gamma$ plays is to make $\mathbf{\Delta} \succ 0$. Hence, in our implementation, we always set $\gamma$ to a very small positive value, such as $10^{-6}$. Actually, we may even set $\gamma$ to 0, because $\mathbf{\Delta}$ does not have to be pd. When $\mathbf{\Delta} \succeq 0$, we say $\mathbf{T}$ follows a singular matrix variate normal distribution [11], and all the derivations for PRPCA are still correct. In our experiment, we find that the performance under $\gamma = 0$ is almost the same as that under $\gamma = 10^{-6}$. Further deliberation is out of the scope of this paper.*

As in PPCA, we set $\mathbf{C} = \mathbf{W}\mathbf{W}^T + \sigma^2\mathbf{I}_d$. Then the log-likelihood of the observation matrix $\mathbf{T}$ in PRPCA is

$$\mathcal{L}_1 = \ln p(\mathbf{T}) = -\frac{N}{2}\Big[d\ln(2\pi) + \ln|\mathbf{C}| + \mathrm{tr}(\mathbf{C}^{-1}\mathbf{H})\Big] + c, \quad (7)$$

where $c = -\frac{d}{2}\ln|\mathbf{\Phi}|$ can be seen as a constant independent of the parameters $\boldsymbol{\mu}$, $\mathbf{W}$ and $\sigma^2$, and $\mathbf{H} = \frac{(\mathbf{T}-\boldsymbol{\mu}\mathbf{e}^T)\mathbf{\Delta}(\mathbf{T}-\boldsymbol{\mu}\mathbf{e}^T)^T}{N}$.

It is interesting to compare (7) with (3). We can find that to learn the parameters $\mathbf{W}$ and $\sigma^2$, the only difference between PRPCA and PPCA lies in the difference between $\mathbf{H}$ and $\mathbf{S}$. Hence, all the learning techniques derived previously for PPCA are also potentially applicable to PRPCA simply by substituting $\mathbf{S}$ with $\mathbf{H}$.

### 4.2 Learning

By setting the gradient of $\mathcal{L}_1$ with respect to $\boldsymbol{\mu}$ to 0, we can get the maximum-likelihood estimator (MLE) for $\boldsymbol{\mu}$ as follows: $\boldsymbol{\mu} = \frac{\mathbf{T}\mathbf{\Delta}\mathbf{e}}{\mathbf{e}^T\mathbf{\Delta}\mathbf{e}}$.

As in PPCA [21], we devise two methods to learn $\mathbf{W}$ and $\sigma^2$ in PRPCA, one based on a closed-form solution and the other based on EM.

### 4.2.1 Closed-Form Solution

**Theorem 1** *The log-likelihood in (7) is maximized when*

$$\mathbf{W}_{ML} = \mathbf{U}_q(\mathbf{\Lambda}_q - \sigma_{ML}^2 \mathbf{I}_q)^{1/2}\mathbf{R}, \qquad \sigma_{ML}^2 = \frac{\sum_{i=q+1}^{d} \lambda_i}{d - q},$$

*where $\lambda_1 \geq \lambda_2 \geq \cdots \geq \lambda_d$ are the eigenvalues of $\mathbf{H}$, $\mathbf{\Lambda}_q$ is a $q \times q$ diagonal matrix containing the first $q$ largest eigenvalues, $\mathbf{U}_q$ is a $d \times q$ matrix in which the $q$ column vectors are the principal eigenvectors of $\mathbf{H}$ corresponding to $\mathbf{\Lambda}_q$, and $\mathbf{R}$ is an arbitrary $q \times q$ orthogonal rotation matrix.*

The proof of Theorem 1 makes use of techniques similar to those in Appendix A of [21] and is omitted here.

### 4.2.2 EM Algorithm

During the EM learning process, we treat $\{\mathbf{W}, \sigma^2\}$ as parameters, $\mathbf{X}$ as missing data and $\{\mathbf{T}, \mathbf{X}\}$ as complete data. The EM algorithm operates by alternating between the E-step and M-step. Here we only briefly describe the updating rules and their derivation can be found in a longer version which can be downloaded from `http://www.cse.ust.hk/~liwujun`.

In the E-step, the expectation of the complete-data log-likelihood with respect to the distribution of the missing data $\mathbf{X}$ is computed. To compute the expectation of the complete-data log-likelihood, we only need to compute the following *sufficient statistics*:

$$\langle \mathbf{X} \rangle = \mathbf{M}^{-1}\mathbf{W}^T(\mathbf{T} - \boldsymbol{\mu}\mathbf{e}^T), \qquad \langle \mathbf{X}\mathbf{\Delta}\mathbf{X}^T \rangle = N\sigma^2\mathbf{M}^{-1} + \langle \mathbf{X} \rangle \mathbf{\Delta} \langle \mathbf{X} \rangle^T, \qquad (8)$$

where $\mathbf{M} = \mathbf{W}^T\mathbf{W} + \sigma^2\mathbf{I}_q$. Note that all these statistics are computed based on the parameter values obtained from the previous iteration.

In the M-step, to maximize the expectation of the complete-data log-likelihood, the parameters $\{\mathbf{W}, \sigma^2\}$ are updated as follows:

$$\widetilde{\mathbf{W}} = \mathbf{H}\mathbf{W}(\sigma^2\mathbf{I}_q + \mathbf{M}^{-1}\mathbf{W}^T\mathbf{H}\mathbf{W})^{-1}, \qquad \widetilde{\sigma}^2 = \frac{\text{tr}(\mathbf{H} - \mathbf{H}\mathbf{W}\mathbf{M}^{-1}\widetilde{\mathbf{W}}^T)}{d}. \qquad (9)$$

Note that we use $\mathbf{W}$ here to denote the old value and $\widetilde{\mathbf{W}}$ for the updated new value.

### 4.3 Complexity Analysis

Suppose there are $\delta$ nonzero elements in $\mathbf{\Delta}$. We can see that the computation cost for $\mathbf{H}$ is $O(dN + d\delta)$. In many applications $\delta$ is typically a constant multiple of $N$. Hence, we can say that the time complexity for computing $\mathbf{H}$ is $O(dN)$. For the closed-form solution, we have to invert a $d \times d$ matrix. Hence, the computation cost is $O(dN + d^3)$. For EM, because $d$ is typically larger than $q$, we can see that the computation cost is $O(dN + d^2qT)$, where $T$ is the number of EM iterations. If the data are of very high dimensionality, EM will be more efficient than the closed-form solution.

## 5 Experiments

Although PPCA possesses additional advantages when compared with the original non-probabilistic formulation of PCA, they will get similar DR results when there exist no missing values in the data. If the task is to classify instances in the low-dimensional embedding, the classifiers based on the embedding results of PCA and PPCA are expected to achieve comparable results. Hence, in this paper, we only adopt PCA as the baseline to study the performance of PRPCA. For the EM algorithm of PRPCA, we use PCA to initialize $\mathbf{W}$, $\sigma^2$ is initialized to $10^{-6}$, and $\gamma = 10^{-6}$. Because the EM algorithm and the closed-form solution achieve similar results, we only report the results of the EM algorithm of PRPCA in the following experiments.

### 5.1 Data Sets and Evaluation Scheme

Here, we only briefly describe the data sets and evaluation scheme for space saving. More detailed information about them can be found in the longer version.

We use three data sets to evaluate PRPCA. The first two data sets are Cora [16] and WebKB [8]. We adopt the same strategy as that in [26] to preprocess these two data sets. The third data set is the PoliticalBook data set used in [19]. For WebKB, according to the semantics of *authoritative pages* and *hub pages* [25], we first preprocess the link structure of this data set as follows: if two web pages are co-linked by or link to another common web page, we add a link between these two pages. Then all the original links are removed. After preprocessing, all the directed links are converted into undirected links.

The Cora data set contains four subsets: DS, HA, ML and PL. The WebKB data set also contains four subsets: Cornell, Texas, Washington and Wisconsin. We adopt the same strategy as that in [26] to evaluate PRPCA on the Cora and WebKB data sets. For the PoliticalBook data set, we use the testing procedure of the latent Wishart process (LWP) model [15] for evaluation.

## 5.2 Convergence Speed of EM

We use the DS and Cornell data sets to illustrate the convergence speed of the EM learning procedure of PRPCA. The performance on other data sets has similar characteristics, which is omitted here. With $q = 50$, the average classification accuracy based on 5-fold cross validation against the number of EM iterations $T$ is shown in Figure 2. We can see that PRPCA achieves very promising and stable performance after a very small number of iterations. We set $T = 5$ in all our following experiments.

## 5.3 Visualization

We use the PoliticalBook data set to visualize the DR results of PCA and PRPCA. For the sake of visualization, $q$ is set to 2. The results are depicted in Figure 3. We can see that it is not easy to separate the two classes in the latent space of PCA. However, the two classes are better separated from each other in the latent space of PRPCA. Hence, better clustering or classification performance can be expected when the examples are clustered or classified in the latent space of PRPCA.

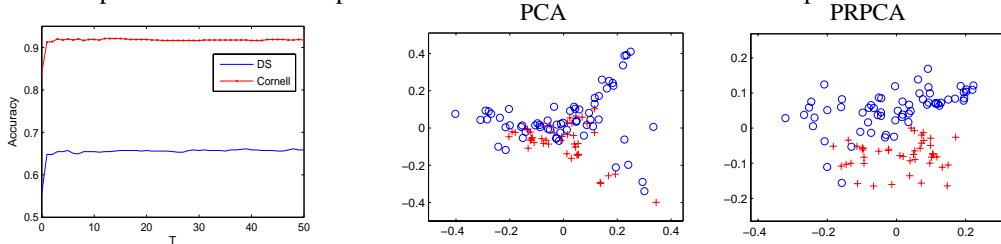

Figure 2: Convergence speed of the EM learning procedure of PRPCA.

Figure 3: Visualization of data points in the latent spaces of PCA and PRPCA for the PoliticalBook data set. The positive and negative examples are shown as red crosses and blue circles, respectively.

## 5.4 Performance

The dimensionality of Cora and WebKB is moderately high, but the dimensionality of PoliticalBook is very high. We evaluate PRPCA on these two different kinds of data to verify its effectiveness in general settings.

**Performance on Cora and WebKB** The average classification accuracy with its standard deviation based on 5-fold cross validation against the dimensionality of the latent space $q$ is shown in Figure 4. We can find that PRPCA can dramatically outperform PCA on all the data sets under any dimensionality, which confirms that the relational information is very informative and PRPCA can utilize it very effectively.

We also perform comparison between PRPCA and those methods evaluated in [26]. The methods include: *SVM on content*, which ignores the link structure in the data and applies SVM only on the content information in the original bag-of-words representation; *SVM on links*, which ignores the content information and treats the links as features, i.e, the $i$th feature is *link-to-page$_i$*; *SVM on link-content*, in which the content features and link features of the two methods above are combined to give the feature representation; *directed graph regularization (DGR)*, which is introduced in [25]; *PLSI+PHITS*, which is described in [7]; *link-content MF*, which is the joint link-content matrix factorization (MF) method in [26]. Note that *Link-content sup. MF* in [26] is not adopted here for comparison. Because during the DR procedure link-content sup. MF employs additional label

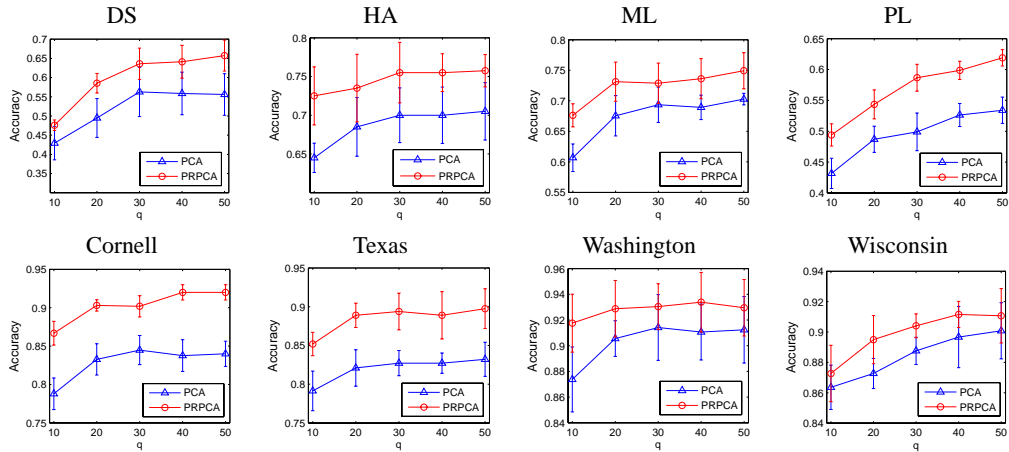
Figure 4: Comparison between PRPCA and PCA on Cora and WebKB.

information which is not employed by other DR methods, it is unfair to directly compare it with other methods. As in the link-content MF method, we set $q = 50$ for PRPCA. The results are shown in Figure 5. We can see that PRPCA and link-content MF achieve the best performance among all the evaluated methods. Compared with link-content MF, PRPCA performs slightly better on DS and HA while performing slightly worse on ML and Texas, and achieves comparable performance on the other data sets. We can conclude that the overall performance of PRPCA is comparable with that of link-content MF. Unlike link-content MF which is transductive in nature, PRPCA naturally supports inductive inference. More specifically, we can apply the learned transformation matrix of PRPCA to perform DR for the unseen test data, while link-content MF can only perform DR for those data available during the training phase. Very recently, another method proposed by us, called *relation regularized matrix factorization* (RRMF) [14], has achieved better performance than PRPCA on the Cora data set. However, similar to link-content MF, RRMF cannot be used for inductive inference either.

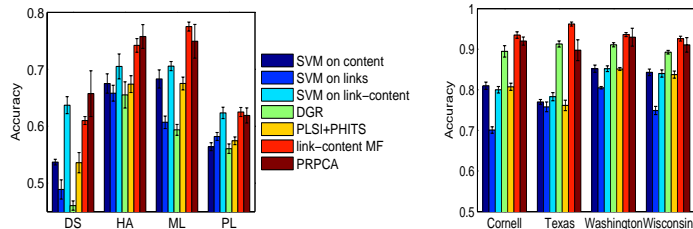
Figure 5: Comparison between PRPCA and other methods on Cora and WebKB.

**Performance on PoliticalBook** As in mixed graph Gaussian process (XGP) [19] and LWP [15], we randomly choose half of the whole data for training and the rest for testing. This subsampling process is repeated for 100 rounds and the average *area under the ROC curve* (AUC) with its standard deviation is reported in Table 1, where GPC is a Gaussian process classifier [18] trained on the original feature representation, and relational Gaussian process (RGP) is the method in [5]. For PCA and PRPCA, we first use them to perform DR, and then a Gaussian process classifier is trained based on the low-dimensional representation. Here, we set $q = 5$ for both PCA and PRPCA. We can see that on this data set, PRPCA also dramatically outperforms PCA and achieves performance comparable with the state of the art. Note that RGP and XGP cannot learn a low-dimensional embedding for the instances. Although LWP can also learn a low-dimensional embedding for the instances, the computation cost to obtain a low-dimensional embedding for a test instance is $O(N^3)$ because it has to invert the kernel matrix defined on the training data.

Table 1: Performance on the PoliticalBook data set. Results for GPC, RGP and XGP are taken from [19] where the standard deviation is not reported.

| GPC | RGP | XGP | LWP | PCA | PRPCA |
|------|------|------|--------------|--------------|--------------|
| 0.92 | 0.98 | 0.98 | $0.98 \pm 0.02$ | $0.92 \pm 0.03$ | $0.98 \pm 0.02$ |

### Acknowledgments

Li and Yeung are supported by General Research Fund 621407 from the Research Grants Council of Hong Kong. Zhang is supported in part by 973 Program (Project No. 2010CB327903). We thank Yu Zhang for some useful comments.

## Footnotes

[1]In this paper, we use document classification as a running example for relational data analysis. Hence, for convenience of illustration, the specific term 'textual content information' is used in the paper to refer to the feature vectors describing the instances. However, the algorithms derived in this paper can be applied to any relational data in which the instance feature vectors can represent any attribute information.

[2]Links with other physical meanings, such as the directed links in web graphs [25], can be transformed into links satisfying the assumption in PRPCA via some preprocessing strategies. One such strategy to preprocess the WebKB data set [8] will be given as an example in Section 5.

[3]This means that we put a 2:1 ratio between $\mathbf{A}$ and $\mathbf{B}$. Other ratios can be obtained by setting $\mathbf{\Delta} = \gamma \mathbf{I}_N + (\alpha \mathbf{I}_N + \mathbf{A})(\alpha \mathbf{I}_N + \mathbf{A}) = \gamma \mathbf{I}_N + \alpha^2 \mathbf{I}_N + 2\alpha \mathbf{A} + \mathbf{B}$. Preliminary results show that PRPCA is not sensitive to $\alpha$ as long as $\alpha$ is not too large, but we omit the detailed results here because they are out of the scope of this paper.

# References

[1] D. J. Bartholomew and M. Knott. *Latent Variable Models and Factor Analysis*. Kendall's Library of Statistics,7, second edition, 1999.

[2] C. M. Bishop. Bayesian PCA. In *NIPS 11*, 1998.

[3] J. Chang and D. M. Blei. Relational topic models for document networks. In *AISTATS*, 2009.

[4] J. Chang, J. L. Boyd-Graber, and D. M. Blei. Connections between the lines: augmenting social networks with text. In *KDD*, pages 169–178, 2009.

[5] W. Chu, V. Sindhwani, Z. Ghahramani, and S. S. Keerthi. Relational learning with Gaussian processes. In *NIPS 19*, 2007.

[6] F. Chung. *Spectral Graph Theory*. Number 92 in Regional Conference Series in Mathematics. American Mathematical Society, 1997.

[7] D. A. Cohn and T. Hofmann. The missing link - a probabilistic model of document content and hypertext connectivity. In *NIPS 13*, 2000.

[8] M. Craven, D. DiPasquo, D. Freitag, A. McCallum, T. M. Mitchell, K. Nigam, and S. Slattery. Learning to extract symbolic knowledge from the world wide web. In *AAAI/IAAI*, pages 509–516, 1998.

[9] A. Dempster, N. Laird, and D. Rubin. Maximum likelihood from incomplete data via the EM algorithm. *Journal of the Royal Statistical Society*, 39(1):1–38, 1977.

[10] L. Getoor and B. Taskar. *Introduction to Statistical Relational Learning*. The MIT Press, 2007.

[11] A. K. Gupta and D. K. Nagar. *Matrix Variate Distributions*. Chapman & Hall/CRC, 2000.

[12] H. Howard. Analysis of a complex of statistical variables into principal components. *Journal of Educational Psychology*, 27:417–441, 1933.

[13] I. T. Jolliffe. *Principal Component Analysis*. Springer, second edition, 2002.

[14] W.-J. Li and D.-Y. Yeung. Relation regularized matrix factorization. In *IJCAI*, 2009.

[15] W.-J. Li, Z. Zhang, and D.-Y. Yeung. Latent Wishart processes for relational kernel learning. In *AISTATS*, pages 336–343, 2009.

[16] A. McCallum, K. Nigam, J. Rennie, and K. Seymore. Automating the construction of internet portals with machine learning. *Information Retrieval*, 3(2):127–163, 2000.

[17] R. Nallapati, A. Ahmed, E. P. Xing, and W. W. Cohen. Joint latent topic models for text and citations. In *KDD*, pages 542–550, 2008.

[18] C. E. Rasmussen and C. K. I. Williams. *Gaussian Processes for Machine Learning*. The MIT Press, 2006.

[19] R. Silva, W. Chu, and Z. Ghahramani. Hidden common cause relations in relational learning. In *NIPS 20*. 2008.

[20] B. Taskar, P. Abbeel, and D. Koller. Discriminative probabilistic models for relational data. In *UAI*, pages 485–492, 2002.

[21] M. E. Tipping and C. M. Bishop. Probabilistic principal component analysis. *Journal Of The Royal Statistical Society Series B*, 61(3):611–622, 1999.

[22] J.-P. Vert. Reconstruction of biological networks by supervised machine learning approaches. In *Elements of Computational Systems Biology*, 2009.

[23] T. Yang, R. Jin, Y. Chi, and S. Zhu. A Bayesian framework for community detection integrating content and link. In *UAI*, 2009.

[24] T. Yang, R. Jin, Y. Chi, and S. Zhu. Combining link and content for community detection: a discriminative approach. In *KDD*, pages 927–936, 2009.

[25] D. Zhou, B. Schölkopf, and T. Hofmann. Semi-supervised learning on directed graphs. In *NIPS 17*, 2004.

[26] S. Zhu, K. Yu, Y. Chi, and Y. Gong. Combining content and link for classification using matrix factorization. In *SIGIR*, 2007.

